# Kernel Regression and Backpropagation Training with Noise

**Petri Koistinen and Lasse Holmström**
Rolf Nevanlinna Institute, University of Helsinki
Teollisuuskatu 23, SF-00510 Helsinki, Finland

## Abstract

One method proposed for improving the generalization capability of a feed-forward network trained with the backpropagation algorithm is to use artificial training vectors which are obtained by adding noise to the original training vectors. We discuss the connection of such backpropagation training with noise to kernel density and kernel regression estimation. We compare by simulated examples (1) backpropagation, (2) backpropagation with noise, and (3) kernel regression in mapping estimation and pattern classification contexts.

## 1  INTRODUCTION

Let $X$ and $Y$ be random vectors taking values in $\mathbf{R}^d$ and $\mathbf{R}^p$, respectively. Suppose that we want to estimate $Y$ in terms of $X$ using a feedforward network whose input-output mapping we denote by $y = g(x, w)$. Here the vector $w$ includes all the weights and biases of the network. Backpropagation training using the quadratic loss (or error) function can be interpreted as an attempt to minimize the expected loss

$$\lambda(w) = E\|g(X, w) - Y\|^2. \tag{1}$$

Suppose that $E\|Y\|^2 < \infty$. Then the regression function

$$m(x) = E[Y|X = x]. \tag{2}$$

minimizes the loss $E\|b(X) - Y\|^2$ over all Borel measurable mappings $b$. Therefore, backpropagation training can also be viewed as an attempt to estimate $m$ with the network $g$.

In practice, one cannot minimize $\lambda$ directly because one does not know enough about the distribution of $(X, Y)$. Instead one minimizes a sample estimate

$$\hat{\lambda}_n(w) = \frac{1}{n} \sum_{i=1}^{n} \|g(X_i, w) - Y_i\|^2 \tag{3}$$

in the hope that weight vectors $w$ that are near optimal for $\hat{\lambda}_n$ are also near optimal for $\lambda$. In fact, under rather mild conditions the minimizer of $\hat{\lambda}_n$ actually converges towards the minimizing set of weights for $\lambda$ as $n \to \infty$, with probability one (White, 1989). However, if $n$ is small compared to the dimension of $w$, minimization of $\hat{\lambda}_n$ can easily lead to overfitting and poor generalization, *i.e.*, weights that render $\hat{\lambda}_n$ small may produce a large expected error $\lambda$.

Many cures for overfitting have been suggested. One can divide the available samples into a training set and a validation set, perform iterative minimization using the training set and stop minimization when network performance over the validation set begins to deteriorate (Holmström et al., 1990, Weigend et al., 1990). In another approach, the minimization objective function is modified to include a term which tries to discourage the network from becoming too complex (Weigend et al., 1990). Network pruning (see, *e.g.*, Sietsma and Dow, 1991) has similar motivation. Here we consider the approach of generating artificial training vectors by adding noise to the original samples. We have recently analyzed such an approach and proved its asymptotic consistency under certain technical conditions (Holmström and Koistinen, 1990).

## 2    ADDITIVE NOISE AND KERNEL REGRESSION

Suppose that we have $n$ original training vectors $(x_i, y_i)$ and want to generate artificial training vectors using additive noise. If the distributions of both $X$ and $Y$ are continuous it is natural to add noise to both $x$ and $y$ components of the sample. However, if the distribution of $X$ is continuous and that of $Y$ is discrete (*e.g.*, in pattern classification), it feels more natural to add noise to the $x$ components only. In Figure 1 we present sampling procedures for both cases. In the $x$-only case the additive noise is generated from a random vector $S_X$ with density $K_X$ whereas in the $x$-and-$y$ case the noise is generated from a random vector $S_{XY}$ with density $K_{XY}$. Notice that we control the magnitude of noise with a scalar smoothing parameter $h > 0$.

In both cases the sampling procedures can be thought of as generating random samples from new random vectors $X_h^{(n)}$ and $Y_h^{(n)}$. Using the same argument as in the Introduction we see that a network trained with the artificial samples tends to approximate the regression function $E[Y_h^{(n)}|X_h^{(n)}]$. Generate $I$ uniformly on $\{1, \ldots, n\}$ and denote by $f$ and $f(\cdot|I = i)$ the density and conditional density of $X_h^{(n)}$. Then in the $x$-only case we get

$$m_h^{(n)}(X_h^{(n)}) := E[Y_h^{(n)}|X_h^{(n)}] = \sum_{i=1}^{n} y_i P(I = i|X_h^{(n)})$$

| **Procedure 1.** | **Procedure 2.** |
|---|---|
| (Add noise to $x$ only) | (Add noise to both $x$ and $y$) |
| 1. Select $i \in \{1, \ldots, n\}$ with equal probability for each index. | 1. Select $i \in \{1, \ldots, n\}$ with equal probability for each index. |
| 2. Draw a sample $s_X$ from density $K_X$ on $\mathbf{R}^d$. | 2. Draw a sample $(s_X, s_Y)$ from density $K_{XY}$ on $\mathbf{R}^{d+p}$. |
| 3. Set $\begin{aligned} x_h^{(n)} &= x_i + h s_X \\ y_h^{(n)} &= y_i. \end{aligned}$ | 3. Set $\begin{aligned} x_h^{(n)} &= x_i + h s_X \\ y_h^{(n)} &= y_i + h s_Y. \end{aligned}$ |

Figure 1: Two Procedures for Generating Artificial Training Vectors.

$$= \sum_{i=1}^{n} y_i \frac{f(X_h^{(n)}|I=i)P(I=i)}{f(X_h^{(n)})} = \sum_{i=1}^{n} y_i \frac{h^{-d}K_X((X_h^{(n)} - x_i)/h) \cdot n^{-1}}{\sum_{j=1}^{n} n^{-1}h^{-d}K_X((X_h^{(n)} - x_i)/h)}.$$

Denoting $K_X$ by $k$ we obtain

$$m_h^{(n)}(x) = \frac{\sum_{i=1}^{n} k((x - x_i)/h)y_i}{\sum_{i=1}^{n} k((x - x_i)/h)}. \tag{4}$$

We result in the same expression also in the $x$-and-$y$ case provided that $\int y K_{XY}(x,y)\,dy = 0$ and that we take $k(x) = \int K_{XY}(x,y)\,dy$ (Watson, 1964). The expression (4) is known as the (Nadaraya-Watson) kernel regression estimator (Nadaraya, 1964, Watson, 1964, Devroye and Wagner, 1980).

A common way to train a $p$-class neural network classifier is to train the network to associate a vector $x$ from class $j$ with the $j$'th unit vector $(0, \ldots, 0, 1, 0, \ldots, 0)$. It is easy to see that then the kernel regression estimator components estimate the class *a posteriori* probabilities using (Parzen-Rosenblatt) kernel density estimators for the class conditional densities. Specht (1990) argues that such a classifier can be considered a neural network. Analogously, a kernel regression estimator can be considered a neural network though such a network would need units proportional to the number of training samples. Recently Specht (1991) has advocated using kernel regression and has also presented a clustering variant requiring only a fixed amount of units. Notice also the resemblance of kernel regression to certain radial basis function schemes (Moody and Darken, 1989, Stokbro et al., 1990).

An often used method for choosing $h$ is to minimize the cross-validated error (Härdle and Marron, 1985, Friedman and Silverman, 1989)

$$M(h) = \frac{1}{n}\sum_{i=1}^{n} \|m_{h,i}^{(n)}(x_i) - y_i\|^2, \qquad m_{h,i}^{(n)}(x) = \frac{\sum_{j \neq i} k((x - x_j)/h)y_j}{\sum_{j \neq i} k((x - x_j)/h)}. \tag{5}$$

Another possibility is to use a method suggested by kernel density estimation theory (Duin, 1976, Habbema et al., 1974) whereby one chooses that $h$ maximizing a cross-validated (pseudo) likelihood function

$$L_{XY}(h) = \prod_{i=1}^{n} f_{n,h,i}^{XY}(x_i, y_i), \qquad L_X(h) = \prod_{i=1}^{n} f_{n,h,i}^{X}(x_i), \tag{6}$$

where $f_{n,h,i}^{XY}$ ($f_{n,h,i}^{X}$) is a kernel density estimate with kernel $K_{XY}$ ($K_X$) and smoothing parameter $h$ but with the $i$'th sample point left out.

## 3    EXPERIMENTS

In the first experiment we try to estimate a mapping $g_0$ from noisy data $(x, y)$,

$$Y = g_0(X) + N_y = a \sin X + b + N_y, \qquad a = 0.4, b = 0.5$$
$$X \sim \text{UNI}(-\pi, \pi), \qquad N_y \sim N(0, \sigma^2), \qquad \sigma = 0.1.$$

Here UNI and $N$ denote the uniform and the normal distribution. We experimented with backpropagation, backpropagation with noise and kernel regression. Backpropagation loss function was minimized using Marquardt's method. The network architecture was FN-1-13-1 with 40 adaptable weights (a feedforward network with one input, 13 hidden nodes, one output, and logistic activation functions in the hidden and output layers). We started the local optimizations from 3 different random initial weights and kept the weights giving the least value for $\hat{\lambda}_n$. Backpropagation training with noise was similar except that instead of the original $n$ vectors we used $10n$ artificial vectors generated with Procedure 2 using $S_{XY} \sim N(0, I_2)$. Magnitude of noise was chosen with the criterion $L_{XY}$ (which, for backpropagation, gave better results than $M$). In the kernel regression experiments $S_{XY}$ was kept the same. Table 1 characterizes the distribution of $J$, the expected squared distance of the estimator $g$ ($g(\cdot, w)$ or $m_h^{(n)}$) from $g_0$,

$$J = E[g(X) - g_0(X)]^2.$$

Table 2 characterizes the distribution of $h$ chosen according to the criteria $L_{XY}$ and $M$ and Figure 2 shows the estimators in one instance. Notice that, on the average, kernel regression is better than backpropagation with noise which is better than plain backpropagation. The success of backpropagation with noise is partly due to the fact that $\sigma$ and $n$ have here been picked favorably. Notice too that in kernel regression the results with the two cross-validation methods are similar although the $h$ values they suggest are clearly different.

In the second experiment we trained classifiers for a four-dimensional two-class problem with equal *a priori* probabilities and class-conditional densities $N(\mu_1, C_1)$ and $N(\mu_2, C_2)$,

$$\mu_1 = 2.32[1\ 0\ 0\ 0]^T, C_1 = I_4; \qquad \mu_2 = 0, C_2 = 4I_4.$$

An FN-4-6-2 with 44 adaptable weights was trained to associate vectors from class 1 with $[0.9\ 0.1]^T$ and vectors from class 2 with $[0.1\ 0.9]^T$. We generated $n/2$ original vectors from each class and a total of $10n$ artificial vectors using Procedure 1 with $S_X \sim N(0, I_4)$. We chose the smoothing parameters, $h_1$ and $h_2$, separately for the two classes using the criterion $L_X$: $h_i$ was chosen by evaluating $L_X$ on class $i$ samples only. We formed separate kernel regression estimators for each class; the $i$'th estimator was trained to output 1 for class $i$ vectors and 0 for the other sample vectors. The $M$ criterion then produces equal values for $h_1$ and $h_2$. The classification rule was to classify $x$ to class $i$ if the output corresponding to the $i$'th class was the maximum output. The error rates are given in Table 3. (The error rate of the Bayesian classifier is 0.116 in this task.) Table 4 summarizes the distribution of $h_1$ and $h_2$ as selected by $L_X$ and $M$.

Table 1: Results for Mapping Estimation. Mean value (left) and standard deviation (right) of $J$ based on 100 repetitions are given for each method.

| $n$ | BP | | BP+noise, $L_{XY}$ | | Kernel regression | | | |
|---|---|---|---|---|---|---|---|---|
| | | | | | $L_{XY}$ | | $M$ | |
| 40 | .0218 | .016 | .0104 | .0079 | .00446 | .0022 | .00365 | .0019 |
| 80 | .00764 | .0048 | .00526 | .0018 | .00250 | .00078 | .00191 | .00077 |

Table 2: Values of $h$ Suggested by the Two Cross-validation Methods in the Mapping Estimation Experiment. Mean value and standard deviation based on 100 repetitions are given.

| $n$ | $L_{XY}$ | | $M$ | |
|---|---|---|---|---|
| 40 | 0.149 | 0.020 | 0.276 | 0.086 |
| 80 | 0.114 | 0.011 | 0.241 | 0.062 |

Table 3: Error Rates for the Different Classifiers. Mean value and standard deviation based on 25 repetitions are given for each method.

| $n$ | BP | | BP+noise, $L_X$ | | Kernel regression | | | |
|---|---|---|---|---|---|---|---|---|
| | | | | | $L_X$ | | $M$ | |
| 44 | .281 | .054 | .189 | .018 | .201 | .022 | .207 | .027 |
| 88 | .264 | .028 | .163 | .011 | .182 | .010 | .184 | .013 |
| 176 | .210 | .023 | .145 | .010 | .164 | .0089 | .164 | .011 |

Table 4: Values of $h_1$ and $h_2$ Suggested by the Two Cross-validation Methods in the Classification Experiment. Mean value and standard deviation based on 25 repetitions are given.

| $n$ | $L_X$ | | | | $M$ | |
|---|---|---|---|---|---|---|
| | $h_1$ | | $h_2$ | | $h_1 = h_2$ | |
| 44 | .818 | .078 | 1.61 | .14 | 1.14 | .27 |
| 88 | .738 | .056 | 1.48 | .11 | 1.01 | .19 |
| 176 | .668 | .048 | 1.35 | .090 | .868 | .10 |

# 4   CONCLUSIONS

Additive noise can improve the generalization capability of a feedforward network trained with the backpropagation approach. The magnitude of the noise cannot be selected blindly, though. Cross-validation-type procedures seem to suit well for the selection of noise magnitude. Kernel regression, however, seems to perform well whenever backpropagation with noise performs well. If the kernel is fixed in kernel regression, we only have to choose the smoothing parameter $h$, and the method is not overly sensitive to its selection.

## References

[Devroye and Wagner, 1980] Devroye, L. and Wagner, T. (1980). Distribution-free consistency results in nonparametric discrimination and regression function estimation. *The Annals of Statistics*, 8(2):231–239.

[Duin, 1976] Duin, R. P. W. (1976). On the choice of smoothing parameters for Parzen estimators of probability density functions. *IEEE Transactions on Computers*, C-25:1175–1179.

[Friedman and Silverman, 1989] Friedman, J. and Silverman, B. (1989). Flexible parsimonious smoothing and additive modeling. *Technometrics*, 31(1):3–21.

[Habbema et al., 1974] Habbema, J. D. F., Hermans, J., and van den Broek, K. (1974). A stepwise discriminant analysis program using density estimation. In Bruckmann, G., editor, *COMPSTAT 1974*, pages 101–110, Wien. Physica Verlag.

[Härdle and Marron, 1985] Härdle, W. and Marron, J. (1985). Optimal bandwidth selection in nonparametric regression function estimation. *The Annals of Statistics*, 13(4):1465–1481.

[Holmström and Koistinen, 1990] Holmström, L. and Koistinen, P. (1990). Using additive noise in back-propagation training. Research Reports A3, Rolf Nevanlinna Institute. To appear in *IEEE Trans. Neural Networks*.

[Holmström et al., 1990] Holmström, L., Koistinen, P., and Ilmoniemi, R. J. (1990). Classification of unaveraged evoked cortical magnetic fields. In *Proc. IJCNN-90-WASH DC*, pages II: 359–362. Lawrence Erlbaum Associates.

[Moody and Darken, 1989] Moody, J. and Darken, C. (1989). Fast learning in networks of locally-tuned processing units. *Neural Computation*, 1:281–294.

[Nadaraya, 1964] Nadaraya, E. (1964). On estimating regression. *Theor. Probability Appl.*, 9:141–142.

[Sietsma and Dow, 1991] Sietsma, J. and Dow, R. J. F. (1991). Creating artificial neural networks that generalize. *Neural Networks*, 4:67–79.

[Specht, 1991] Specht, D. (1991). A general regression neural network. *IEEE Transactions on Neural Networks*, 2(6):568–576.

[Specht, 1990] Specht, D. F. (1990). Probabilistic neural networks. *Neural Networks*, 3(1):109–118.

[Stokbro et al., 1990] Stokbro, K., Umberger, D., and Hertz, J. (1990). Exploiting neurons with localized receptive fields to learn chaos. NORDITA preprint.

[Watson, 1964] Watson, G. (1964). Smooth regression analysis. *Sankhyā Ser. A*, 26:359–372.

[Weigend et al., 1990] Weigend, A., Huberman, B., and Rumelhart, D. (1990). Predicting the future: A connectionist approach. *International Journal of Neural Systems*, 1(3):193–209.

[White, 1989] White, H. (1989). Learning in artificial neural networks: A statistical perspective. *Neural Computation*, 1:425–464.

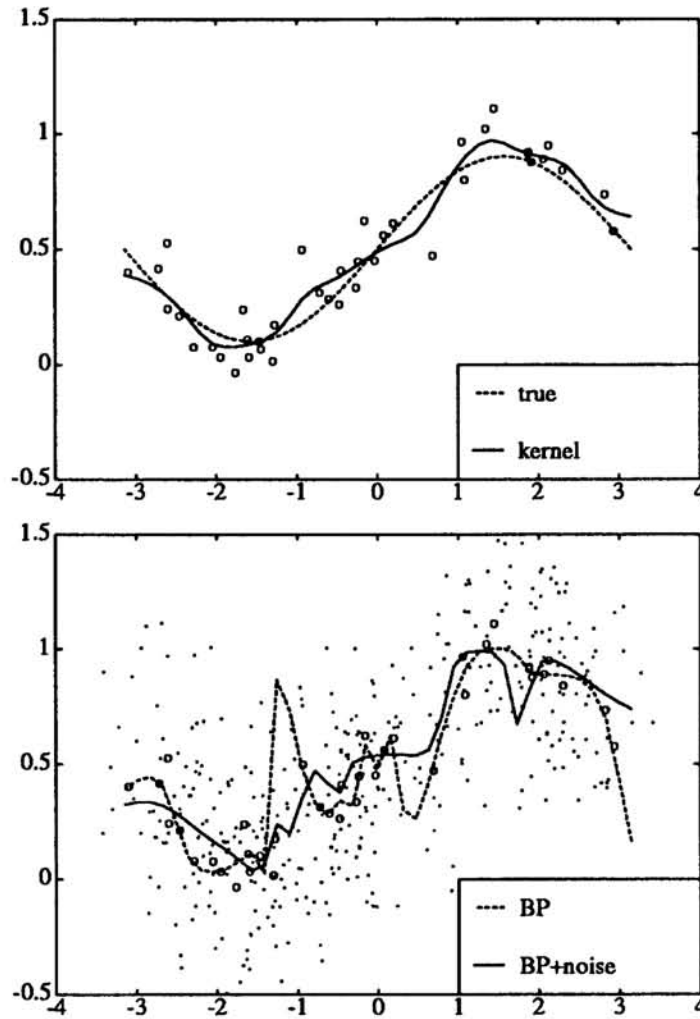

Figure 2: Results From a Mapping Estimation Experiment. Shown are the $n = 40$ original vectors (o's), the artificial vectors (dots), the true function $a \sin x + b$ and the fitting results using kernel regression, backpropagation and backpropagation with noise. Here $h = 0.16$ was chosen with $L_{XY}$. Values of $J$ are 0.0075 (kernel regression), 0.014 (backpropagation with noise) and 0.038 (backpropagation).